# Manifold Denoising

**Matthias Hein**      **Markus Maier**
Max Planck Institute for Biological Cybernetics
Tübingen, Germany
{first.last}@tuebingen.mpg.de

## Abstract

We consider the problem of denoising a noisily sampled submanifold $M$ in $\mathbb{R}^d$, where the submanifold $M$ is a priori unknown and we are only given a noisy point sample. The presented denoising algorithm is based on a graph-based diffusion process of the point sample. We analyze this diffusion process using recent results about the convergence of graph Laplacians. In the experiments we show that our method is capable of dealing with non-trivial high-dimensional noise. Moreover using the denoising algorithm as pre-processing method we can improve the results of a semi-supervised learning algorithm.

## 1 Introduction

In the last years several new methods have been developed in the machine learning community which are based on the assumption that the data lies on a submanifold $M$ in $\mathbb{R}^d$. They have been used in semi-supervised learning [15], dimensionality reduction [14, 1] and clustering. However there exists a certain gap between theory and practice. Namely in practice the data lies almost never exactly on the submanifold but due to noise is scattered around it. Several of the existing algorithms in particular graph based methods are quite sensitive to noise. Often they fail in the presence of high-dimensional noise since then the distance structure is non-discriminative. In this paper we tackle this problem by proposing a denoising method for manifold data. Given noisily sampled manifold data in $\mathbb{R}^d$ the objective is to 'project' the sample onto the submanifold.

There exist already some methods which have related objectives like principal curves [6] and the generative topographic mapping [2]. For both methods one has to know the intrinsic dimension of the submanifold $M$ as a parameter of the algorithm. However in the presence of high-dimensional noise it is almost impossible to estimate the intrinsic dimension correctly. Moreover usually problems arise if there is more than one connected component.

The algorithm we propose adresses these problems. It works well for low-dimensional submanifolds corrupted by high-dimensional noise and can deal with multiple connected components. The basic principle behind our denoising method has been proposed by [13] as a surface processing method in $\mathbb{R}^3$. The goal of this paper is twofold. First we extend this method to general submanifolds in $\mathbb{R}^d$ aimed at dealing in particular with high-dimensional noise. Second we provide an interpretation of the denoising algorithm which takes into account the probabilistic setting encountered in machine learning and which differs from the one usually given in the computer graphics community.

## 2 The noise model and problem statement

We assume that the data lies on an abstract $m$-dimensional manifold $M$, where the dimension $m$ can be seen as the number of independent parameters in the data. This data is mapped via a smooth, regular embedding $i : M \to \mathbb{R}^d$ into the feature space $\mathbb{R}^d$. In the following we will not distinguish between $M$ and $i(M) \subset \mathbb{R}^d$, since it should be clear from the context which case we are considering. The Euclidean distance in $\mathbb{R}^d$ then induces a metric on $M$. This metric depends on the

embedding/representation (e.g. scaling) of the data in $\mathbb{R}^d$ but is at least continuous with respect to the intrinsic parameters. Furthermore we assume that the manifold $M$ is equipped with a probability measure $P_M$ which is absolutely continuous with respect to the natural volume element[1] $dV$ of $M$. With these definitions the model for the noisy data-generating process in $\mathbb{R}^d$ has the following form:

$$X = i(\Theta) + \epsilon,$$

where $\Theta \sim P_M$ and $\epsilon \sim N(0, \sigma)$. Note that the probability measure of the noise $\epsilon$ has full support in $\mathbb{R}^d$. We consider here for convenience a Gaussian noise model but also any other reasonably concentrated isotropic noise should work. The law $P_X$ of the noisy data $X$ can be computed from the true data-generating probability measure $P_M$:

$$P_X(x) = (2\pi\sigma^2)^{-\frac{d}{2}} \int_M e^{-\frac{\|x-i(\theta)\|^2}{2\sigma^2}} p(\theta)\, dV(\theta). \tag{1}$$

Now the Gaussian measure is equivalent to the heat kernel $p_t(x,y) = (4\pi t)^{-\frac{d}{2}} \exp\left(-\frac{\|x-y\|^2}{4t}\right)$ of the diffusion process on $\mathbb{R}^d$, see e.g. [5], if we make the identification $\sigma^2 = 2t$. An alternative point of view on $P_X$ is therefore to see $P_X$ as the result of a diffusion of the density function[2] $p(\theta)$ of $P_M$ stopped at time $t = \frac{1}{2}\sigma^2$. The basic principle behind the denoising algorithm in this paper is to *reverse* this diffusion process.

## 3 The denoising algorithm

In practice we have only an i.i.d. sample $X_i$, $i = 1, \ldots, n$ of $P_X$. The ideal goal would be to find the corresponding set of points $i(\theta_i)$, $i = 1, \ldots, n$ on the submanifold $M$ which generated the points $X_i$. However due to the random nature of the noise this is in principle impossible. Instead the goal is to find corresponding points $Z_i$ on the submanifold $M$ which are close to the points $X_i$. However we are facing several problems. Since we are only given a finite sample, we do not know $P_X$ or even $P_M$. Second as stated in the last section we would like to reverse this diffusion process which amounts to solving a PDE. However the usual technique to solve this PDE on a grid is unfeasible due to the high dimension of the ambient space $\mathbb{R}^d$.

Instead we solve the diffusion process directly on a graph generated by the sample $X_i$. This can be motivated by recent results in [7] where it was shown that the generator of the diffusion process, the Laplacian $\Delta_{\mathbb{R}^d}$, can be approximated by the graph Laplacian of a random neighborhood graph. A similar setting for the denoising of two-dimensional meshes in $\mathbb{R}^3$ has been proposed in the seminal work of Taubin [13]. Since then several modifications of his original idea have been proposed in the computer graphics community, including the recent development in [11] to apply the algorithm directly to point cloud data in $\mathbb{R}^3$. In this paper we propose a modification of this diffusion process which allows us to deal with general noisy samples of arbitrary (low-dimensional) submanifolds in $\mathbb{R}^d$. In particular the proposed algorithm can cope with high-dimensional noise. Moreover we give an interpretation of the algorithm, which differs from the one usually given in the computer graphics community and takes into account the probabilistic nature of the problem.

### 3.1 Structure on the sample-based graph

We would like to define a diffusion process directly on the sample $X_i$. To this end we need the generator of the diffusion process, the graph Laplacian. We will construct this operator for a weighted, undirected graph. The graph vertices are the sample points $X_i$. With $\{h(X_i)\}_{i=1}^n$ being the $k$-nearest neighbor ($k$-NN) distances the weights of the $k$-NN graph are defined as

$$w(X_i, X_j) = \exp\left(-\frac{\|X_i - X_j\|^2}{(\max\{h(X_i), h(X_j)\})^2}\right), \qquad \text{if} \quad \|X_i - X_j\| \le \max\{h(X_i), h(X_j)\},$$

and $w(X_i, X_j) = 0$ otherwise. Additionally we set $w(X_i, X_i) = 0$, so that the graph has no loops. Further we denote by $d$ the degree function $d(X_i) = \sum_{j=1}^n w(X_i, X_j)$ of the graph and

we introduce two Hilbert spaces $\mathcal{H}_V, \mathcal{H}_E$ of functions on the vertices $V$ and edges $E$. Their inner products are defined as

$$\langle f, g \rangle_{\mathcal{H}_V} = \sum\nolimits_{i=1}^{n} f(X_i)\, g(X_i)\, d(X_i), \qquad \langle \phi, \psi \rangle_{\mathcal{H}_E} = \sum\nolimits_{i,j=1}^{n} w(X_i, X_j)\, \phi(X_i, X_j)\, \psi(X_i, X_j).$$

Introducing the discrete differential $\nabla : \mathcal{H}_V \to \mathcal{H}_E$, $(\nabla f)(X_i, X_j) = f(X_j) - f(X_i)$ the graph Laplacian is defined as

$$\Delta : \mathcal{H}_V \to \mathcal{H}_V, \quad \Delta = \nabla^* \nabla, \quad (\Delta f)(X_i) = f(X_i) - \frac{1}{d(X_i)} \sum\nolimits_{j=1}^{n} w(X_i, X_j) f(X_j),$$

where $\nabla^*$ is the adjoint of $\nabla$. Defining the matrix $D$ with the degree function on the diagonal the graph Laplacian in matrix form is given as $\Delta = \mathbb{1} - D^{-1}W$, see [7] for more details. Note that despite $\Delta$ is not a symmetric matrix it is a self-adjoint operator with respect to the inner product in $\mathcal{H}_V$.

## 3.2 The denoising algorithm

Having defined the necessary structure on the graph it is straightforward to write down the backward diffusion process. In the next section we will analyze the geometric properties of this diffusion process and show why it is directed towards the submanifold $M$. Since the graph Laplacian is the generator of the diffusion process on the graph we can formulate the algorithm by the following differential equation on the graph:

$$\partial_t X = -\gamma\, \Delta X, \tag{2}$$

where $\gamma > 0$ is the diffusion constant. Since the points change with time, the whole graph is dynamic in our setting. This is different to the diffusion processes on a fixed graph studied in semi-supervised learning. In order to solve the differential equation (2) we choose an implicit Euler-scheme, that is

$$X(t+1) - X(t) = -\delta t\, \gamma\, \Delta X(t+1), \tag{3}$$

where $\delta t$ is the time-step. Since the implicit Euler is unconditionally stable we can choose the factor $\delta t\, \gamma$ arbitrarily. We fix in the following $\gamma = 1$ so that the only free parameter remains to be $\delta t$, which is set to $\delta = 0.5$ in the rest of the paper. The solution of the implicit Euler scheme for one timestep in Equation 3 can then be computed as: $X_{t+1} = (\mathbb{1} + \delta t\, \Delta)^{-1} X_t$. After each timestep the point configuration has changed so that one has to recompute the weight matrix $W$ of the graph. Then the procedure is continued until a predefined stopping criterion is satisfied, see Section 3.4. The pseudo-code is given in Algorithm 1. In [12] it was pointed out that there exists a connection

---

**Algorithm 1** Manifold denoising

1: Choose $\delta t, k$
2: **while** Stopping criterion not satisfied **do**
3:     Compute the $k$-NN distances $h(X_i)$, $i = 1, \ldots, n$,
4:     Compute the weights $w(X_i, X_j)$ of the graph with $w(X_i, X_i) = 0$,
$$w(X_i, X_j) = \exp\left(-\frac{\|X_i - X_j\|^2}{(\max\{h(X_i), h(X_j)\})^2}\right), \quad \text{if} \quad \|X_i - X_j\| \leq \max\{h(X_i), h(X_j)\},$$
5:     Compute the graph Laplacian $\Delta$,    $\Delta = \mathbb{1} - D^{-1}W$,
6:     Solve $X(t+1) - X(t) = -\delta t\, \Delta X(t+1) \quad \Rightarrow \quad X(t+1) = (\mathbb{1} + \delta t\, \Delta)^{-1} X(t)$.
7: **end while**

---

between diffusion processes and Tikhonov regularization. Namely the result of one time step of the diffusion process with the implicit Euler scheme is equivalent to the solution of the following regularization problem on the graph:

$$\underset{Z^\alpha \in \mathcal{H}_V}{\arg\min}\, \mathrm{S}(Z^\alpha) := \underset{Z^\alpha \in \mathcal{H}_V}{\arg\min} \sum_{\alpha=1}^{d} \|Z^\alpha - X^\alpha(t)\|_{\mathcal{H}_V}^2 + \delta t \sum_{\alpha=1}^{d} \|\nabla Z^\alpha\|_{\mathcal{H}_E}^2,$$

where $Z^\alpha$ denotes the $\alpha$-component of the vector $Z \in \mathbb{R}^d$. With $\|\nabla Z^\alpha\|_{\mathcal{H}_E}^2 = \langle Z^\alpha, \Delta\, Z^\alpha \rangle_{\mathcal{H}_V}$ the minimizer of the above functional with respect to $Z^\alpha$ can be easily computed as

$$\frac{\partial \mathrm{S}(Z^\alpha)}{\partial Z^\alpha} = 2(Z^\alpha - X^\alpha(t)) + 2\,\delta t\, \Delta Z^\alpha = 0, \quad \alpha = 1, \ldots, d,$$

so that $Z = (\mathbb{1} + \delta t\,\Delta)^{-1}X_t$. Each time-step of our diffusion process can therefore be seen as a regression problem, where we trade off between fitting the new points $Z$ to the points $X(t)$ and having a 'smooth' point configuration $Z$ measured with respect to the current graph built from $X(t)$.

### 3.3 $k$-nearest neighbor graph versus $h$-neighborhood graph

In the denoising algorithm we have chosen to use a weighted $k$-NN graph. It turns out that a $k$-NN graph has three advantages over an $h$-neighborhood graph[3]. The first advantage is that the graph has a better connectivity. Namely points in areas of different density have quite different neighborhood scales which leads for a fixed $h$ to either disconnected or over-connected graphs.
Second we usually have high-dimensional noise. In this case it is well-known that one has a drastic change in the distance statistic of a sample, which is illustrated by the following trivial lemma.

**Lemma 1** *Let* $x, y \in \mathbb{R}^d$ *and* $\epsilon_1, \epsilon_2 \sim N(0, \sigma^2)$ *and define* $X = x + \epsilon_1$ *and* $Y = y + \epsilon_2$, *then*

$$\mathbb{E}\,\|X - Y\|^2 = \|x - y\|^2 + 2\,d\,\sigma^2, \quad \text{and} \quad \operatorname{Var}\|X - Y\|^2 = 8\sigma^2\,\|x - y\|^2 + 8\,d\,\sigma^4.$$

One can deduce that the expected squared distance of the noisy submanifold sample is dominated by the noise term if $2d\sigma^2 > \max_{\theta, \theta'}\|i(\theta) - i(\theta')\|^2$, which is usually the case for large $d$. In this case it is quite difficult to adjust the average number of neighbors in a graph by a fixed neighborhood size $h$ since the distances start to concentrate around their mean value. The third is that by choosing $k$ we can control directly the sparsity of the weight matrix $W$ and the Laplacian $\Delta = \mathbb{1} - D^{-1}W$ so that the linear equation in each time step can be solved efficiently.

### 3.4 Stopping criterion

The problem of choosing the correct number of iterations is very difficult if one has initially high-dimensional noise and requires prior knowledge. We propose two stopping criterions. The first one is based on the effect that if the diffusion is done too long the data becomes disconnected and concentrates in local clusters. One therefore can stop if the number of connected components of the graph[4] increases. The second one is based on prior knowledge about the intrinsic dimension of the data. In this case one can stop the denoising if the estimated dimension of the sample (e.g. via the correlation dimension, see [4]) is equal to the intrinsic one. Another less founded but very simple way is to stop the iterations if the changes in the sample are below some pre-defined threshold.

## 4 Large sample limit and theoretical analysis

Our qualitative theoretical analysis of the denoising algorithm is based on recent results on the limit of graph Laplacians [7, 8] as the neighborhood size decreases and the sample size increases. We use this result to study the continuous limit of the diffusion process. The following theorem about the limit of the graph Laplacian applies to $h$-neighborhood graphs, whereas the denoising algorithm is based on a $k$-NN graph. Our conjecture[5] is that the result carries over to $k$-NN graphs.

**Theorem 1** *[7, 8] Let* $\{X_i\}_{i=1}^n$ *be an i.i.d. sample of a probability measure* $P_M$ *on a* $m$-*dimensional compact submanifold[6]* $M$ *of* $\mathbb{R}^d$, *where* $P_M$ *has a density* $p_M \in C^3(M)$. *Let* $f \in C^3(M)$ *and* $x \in M \backslash \partial M$, *then if* $h \to 0$ *and* $nh^{m+2}/\log n \to \infty$,

$$\lim_{n \to \infty} \frac{1}{h^2}(\Delta f)(x) \sim -(\Delta_M f)(x) - \frac{2}{p}\,\langle \nabla f, \nabla p \rangle_{T_x M}, \quad almost \;\; surely,$$

*where* $\Delta_M$ *is the Laplace-Beltrami operator of* $M$ *and* $\sim$ *means up to a constant which depends on the kernel function* $k(\|x - y\|)$ *used to define the weights* $W(x, y) = k(\|x - y\|)$ *of the graph.*

## 4.1 The noise-free case

We first derive in a non-rigorous way the continuum limit of our graph based diffusion process in the noise free case. To that end we do the usual argument made in physics to go from a difference equation on a grid to the differential equation. We rewrite our diffusion equation (2) on the graph as

$$\frac{i(t+1)-i(t)}{\delta t} = -\frac{h^2}{\delta t}\frac{1}{h^2}\Delta i$$

Doing now the limit $h \to 0$ and $\delta t \to 0$ such that the diffusion constant $D = \frac{h^2}{\delta t}$ stays finite and using the limit of $\frac{1}{h^2}\Delta$ given in Theorem 1 we get the following differential equation,

$$\partial_t i = D\left[\Delta_M i + \frac{2}{p}\langle \nabla p, \nabla i\rangle\right]. \tag{4}$$

Note that for the $k$-NN graph the neighborhood size $h$ is a function of the local density which implies that the diffusion constant $D$ also becomes a function of the local density $D = D(p(x))$.

**Lemma 2 ([9], Lemma 2.14)** *Let* $i : M \to \mathbb{R}^d$ *be a regular, smooth embedding of an $m$-dimensional manifold $M$, then $\Delta_M i = m\,H$, where $H$ is the mean curvature[7] of $M$.*

Using the equation $\Delta_M i = mH$ we can establish equivalence of the continuous diffusion equation (4) to a generalized mean curvature flow.

$$\partial_t i = D\left[m\,H + \frac{2}{p}\langle \nabla p, \nabla i\rangle\right], \tag{5}$$

The equivalence to the mean curvature flow $\partial_t i = m\,H$ is usually given in computer graphics as the reason for the denoising effect, see [13, 11]. However as we have shown the diffusion has already an additional part if one has a non-uniform probability measure on $M$.

## 4.2 The noisy case

The analysis of the noisy case is more complicated and we can only provide a rough analysis. The large sample limit $n \to \infty$ of the graph Laplacian $\Delta$ at a sample point $X_i$ is given as

$$\Delta X_i = X_i - \frac{\int_{\mathbb{R}^d} k_h(\|X_i - y\|)\,y\,p_X(y)dy}{\int_{\mathbb{R}^d} k_h(\|X_i - y\|)p_X(y)dy}, \tag{6}$$

where $k_h(\|x-y\|)$ is the weight function used in the construction of the graph, that is in our case $k_h(\|x-y\|) = e^{-\frac{\|x-y\|^2}{2h^2}}\mathbb{1}_{\|x-y\|\le h}$. In the following analysis we will assume three things, 1) the noise level $\sigma$ is small compared to the neighborhood size $h$, 2) the curvature of $M$ is small compared to $h$ and 3) the density $p_M$ varies slowly along $M$. Under these conditions it is easy to see that the main contribution of $-\Delta X_i$ in Equation 6 will be in the direction of the gradient of $p_X$ at $X_i$. In the following we try to separate this effect from the mean curvature part derived in the noise-free case. Under the above conditions we can do the following second order approximation of a convolution with a Gaussian, see [7], using the explicit form of $p_X$ of Equation 1 :

$$\int_{\mathbb{R}^d} k_h(\|X-y\|)\,y\,p_X(y)dy = \int_M \frac{1}{(2\pi\sigma^2)^{d/2}}\int_{\mathbb{R}^d} k_h(\|X-y\|)\,y\,e^{-\frac{\|y-i(\theta)\|^2}{2\sigma^2}}\,p(\theta)\,dy\,dV(\theta)$$

$$= \int_M k_h(\|X-i(\theta)\|)\,i(\theta)\,p(\theta)\,dV(\theta) + O(\sigma^2)$$

Now define the closest point of the submanifold $M$ to $X$: $i(\theta_{\min}) = \arg\min_{i(\theta)\in M}\|X-i(\theta)\|$. Using the condition on the curvature we can approximate the diffusion step $-\Delta X$ as follows:

$$-\Delta X \approx \underbrace{i(\theta_{\min}) - X}_{I} - \left(\underbrace{i(\theta_{\min}) - \frac{\int_M k_h(\|i(\theta_{\min})-i(\theta)\|)\,i(\theta)\,p(\theta)\,dV(\theta)}{\int_M k_h(\|i(\theta_{\min})-i(\theta)\|)\,p(\theta)\,dV(\theta)}}_{II}\right),$$

where we have omitted second-order terms. It follows from the proof of Theorem 1 that the term $II$ is an approximation of $-\Delta_M i(\theta_{\min}) - \frac{2}{p}\langle\nabla p, \nabla i\rangle = -mH - \frac{2}{p}\langle\nabla p, \nabla i\rangle$ whereas the first term $I$ leads to a movement of $X$ towards $M$. We conclude from this rough analysis that in the denoising procedure we always have a tradeoff between reducing the noise via the term $I$ and smoothing of the manifold via the mean curvature term $II$. Note that the term $II$ is the same for all points $X$ which have $i(\theta_{\min})$ as their closest point on $M$. Therefore this term leads to a global flow which smoothes the submanifold. In the experiments we observe this as the shrinking phenomenon.

## 5  Experiments

In the experimental section we test the performance of the denoising algorithm on three noisy datasets. Furthermore we explore the possibility to use the denoising method as a preprocessing step for semi-supervised learning. Due to lack of space we can not deal with further applications as preprocessing method for clustering or dimensionality reduction.

### 5.1  Denoising

The first experiment is done on a toy-dataset. The manifold $M$ is given as $t \rightarrow [\sin(2\pi t), \ 2\pi t]$, $t$ is sampled uniformly on $[0, 1]$. We embed $M$ into $\mathbb{R}^{200}$ and put full isotropic Gaussian noise with $\sigma = 0.4$ on each datapoint resulting in the left part of Figure 5.1. We verify the effect of the denoising algorithm by estimating continuously the dimension over different scales (note that the dimension of a finite sample always depends on the scale at which one examines). We use for that purpose the correlation dimension estimator of [4].

The result of the denoising algorithm with $k = 25$ for the $k$-NN graph and 10 timesteps is given in the right part of Figure 5.1. One can observe visually and by inspecting the dimension estimate as well as by the histogram of distances that the algorithm has reduced the noise. One can also see two undesired effects. First as discussed in the last section the diffusion process has a component which moves the manifold in the direction of the mean curvature, which leads to a smoothing of the sinusoid. Second at the boundary the sinusoid shrinks due to the missing counterparts in the local averaging done by the graph Laplacian, see (6), which result in an inward tangential component.

In the next experiment we apply the denoising to the handwritten digit datasets USPS and MNIST.

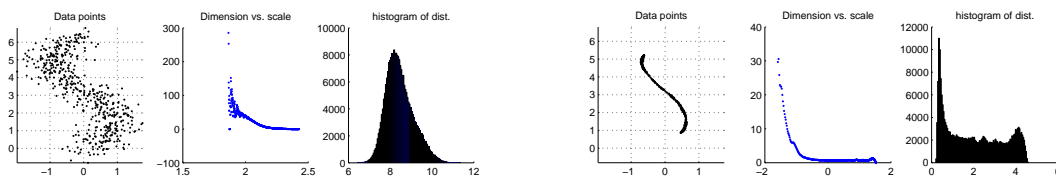

Figure 1: Left: 500 samples of the noisy sinusoid in $\mathbb{R}^{200}$ as described in the text, Right: Result after 10 steps of the denoising method with $k = 25$, note that the estimated dimension is much smaller and the scale has changed as can be seen from the histogram of distances shown to the right

For handwritten digits the underlying manifold corresponds to varying writing styles. In order to check if the denoising method can also handle several manifolds at the same time which would make the method useful for clustering and dimensionality reduction we fed all the 10 digits simultaneously into the algorithm. For USPS we used the 9298 digits in the training and test set and from MNIST a subsample of 1000 examples from each digit. We used the two-sided tangent distance in [10] which provides a certain invariance against translation, scaling, rotation and line thickness. In Figure 2 and 3 we show a sample of the result across all digits. In both cases digits are transformed wrongly. This happens since they are outliers with respect to their digit manifold and lie closer to another digit component. An improved handling of invariances should resolve at least partially this problem.

### 5.2  Denoising as pre-processing for semi-supervised learning

Most semi-supervised learning (SSL) are based on the cluster assumption, that is the decision boundary should lie in a low-density region. The denoising algorithm is consistent with that assumption

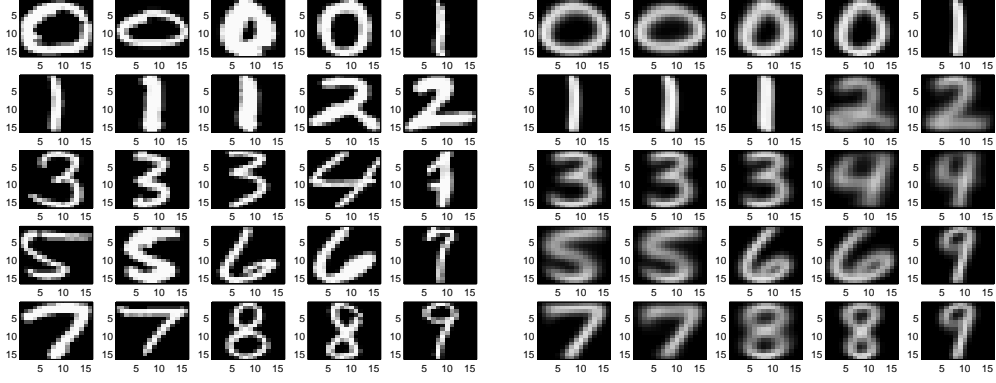

Figure 2: Left: Original images from USPS, right: after 15 iterations with $k = [9298/50]$.

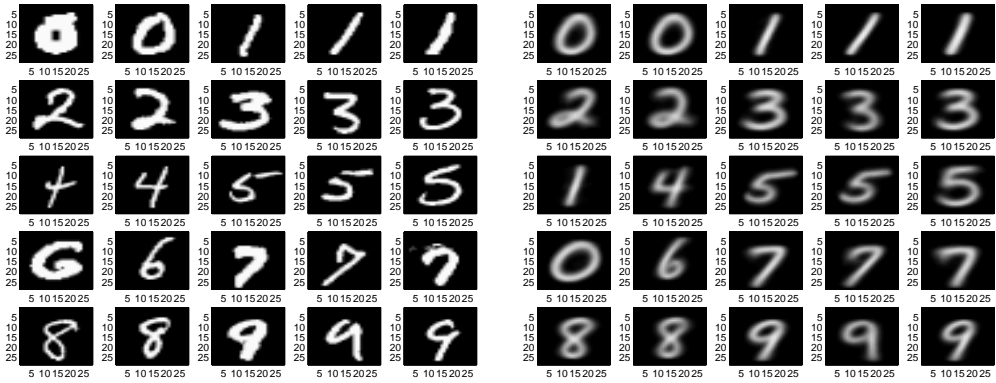

Figure 3: Left: Original images from MNIST, right: after 15 iterations with $k = 100$.

since it moves data points towards high-density regions. This is in particular helpful if the original clusters are distorted by high-dimensional noise. In this case the distance structure of the data becomes less discriminative, see Lemma 1, and the identification of the low density regions is quite difficult. We expect that in such cases manifold denoising as a pre-processing step should improve the discriminative capacity of graph-based methods. However the denoising algorithm does not take into account label information. Therefore in the case where the cluster assumption is not fulfilled the denoising algorithm might decrease the performance. Therefore we add the number of iterations of the denoising process as an additional parameter in the SSL algorithm.

For the evaluation of our denoising algorithm as a preprocessing step for SSL, we used the benchmark data sets from [3]. A description of the data sets and the results of several state-of-the-art SSL algorithms can be found there. As SSL-algorithm we use a slight variation of the one by Zhou et al. [15]. It can be formulated as the following regularized least squares problem.

$$f^* = \mathrm{argmin}_{f \in \mathcal{H}_V} \|f - y\|^2_{\mathcal{H}_V} + \mu \langle f, \Delta f \rangle_{\mathcal{H}_V},$$

where $y$ is the given label vector and $\langle f, \Delta f \rangle_{\mathcal{H}_V}$ is the smoothness functional induced by the graph Laplacian. The solution is given as $f^* = (\mathbb{1} + \mu\Delta)^{-1}y$. In order to be consistent with our denoising scheme we choose instead of the normalized graph Laplacian $\tilde{\Delta} = \mathbb{1} - D^{-\frac{1}{2}}WD^{-\frac{1}{2}}$ as suggested in [15] the graph Laplacian $\Delta = \mathbb{1} - D^{-1}W$ and the graph structure as described in Section 3.1. As neighborhood graph for the SSL-algorithm we used a symmetric $k$-NN graph with the following weights: $w(X_i, X_j) = \exp(-\gamma \|X_i - X_j\|^2)$ if $\|X_i - X_j\| \leq \min\{h(X_i), h(X_j)\}$. As suggested in [3] the distances are rescaled in each iteration such that the $1/c^2$-quantile of the distances equals 1 where $c$ is the number of classes. The number of $k$-NN was chosen for denoising in $\{5, 10, 15, 25, 50, 100, 150, 200\}$, and for classification in $\{5, 10, 20, 50, 100\}$. The scaling

parameter $\gamma$ and the regularization parameter $\mu$ were selected from $\{\frac{1}{2}, 1, 2\}$ resp. $\{2, 20, 200\}$. The maximum of iterations was set to 20. Parameter values where not all data points have been classified, that is the graph is disconnected, were excluded. The best parameters were found by ten-fold cross validation. The final classification is done using a majority vote of the classifiers corresponding to the minimal cross validation test error. In Table 1 the results are shown for the standard case, that is no manifold denoising (No MD), and with manifold denoising (MD). For the datasets g241c, g241d and Text we get significantly better performance using denoising as a preprocessing step, whereas the results are indifferent for the other datasets. However compared to the results of the state of the art of SSL on all the datasets reported in [3], the denoising preprocessing has lead to a performance of the algorithm which is competitive uniformly over all datasets. This improvement is probably not limited to the employed SSL-algorithm but should also apply to other graph-based methods.

Table 1: Manifold Denoising (MD) as preprocessing for SSL. The mean and standard deviation of the test error are shown for the datasets from [3] for 10 (top) and 100 (bottom) labeled points.

|         | g241c     | g241d     | Digit1   | USPS     | COIL     | BCI      | Text     |
|---------|-----------|-----------|----------|----------|----------|----------|----------|
| No MD   | 47.9±2.67 | 47.2±4.0  | 14.1±5.4 | 19.2±2.1 | 66.2±7.8 | 50.0±1.1 | 41.9±7.0 |
| MD      | 29.0±14.3 | 26.6±17.8 | 13.8±5.5 | 20.5±5.0 | 66.4±6.0 | 49.8±1.5 | 33.6±7.0 |
| ø Iter. | 12.3±3.8  | 11.7±4.4  | 9.6±2.4  | 7.3±2.9  | 4.9±2.7  | 8.2±3.5  | 5.6±4.4  |
| No MD   | 38.9±6.3  | 34.2±4.1  | 3.0±1.6  | 6.2±1.2  | 15.5±2.6 | 46.5±1.9 | 27.0±1.9 |
| MD      | 16.1±2.2  | 7.5±0.9   | 3.2±1.2  | 5.3±1.4  | 16.2±2.5 | 48.4±2.0 | 24.1±2.8 |
| ø Iter. | 15.0±0.8  | 14.5±1.5  | 8.0±3.2  | 8.3±3.8  | 1.6±1.8  | 8.4±4.3  | 6.0±3.5  |

## Footnotes

[1] In local coordinates $\theta_1, \ldots, \theta_m$ the natural volume element $dV$ is given as $dV = \sqrt{\det g}\, d\theta_1 \ldots d\theta_m$, where $\det g$ is the determinant of the metric tensor $g$.

[2] Note that $P_M$ is not absolutely continuous with respect to the Lebesgue measure in $\mathbb{R}^d$ and therefore $p(\theta)$ is not a density in $\mathbb{R}^d$.

[3]In an $h$-neighborhood graph two sample points $X_i$, $X_j$ have a common edge if $\|X_i - X_j\| \leq h$.

[4]The number of connected comp. is equal to the multiplicity of the first eigenvalue of the graph Laplacian.

[5]Partially we verified the conjecture however the proof would go beyond the scope of this paper.

[6]Note that the case where $P$ has full support in $\mathbb{R}^d$ is a special case of this theorem.

[7]The mean curvature $H$ is the trace of the second fundamental form. If $M$ is a hypersurface in $\mathbb{R}^d$ the mean curvature at $p$ is $H = \frac{1}{d-1}\sum_{i=1}^{d-1}\kappa_i N$, where $N$ is the normal vector and $\kappa_i$ the principal curvatures at $p$.

# References

[1] M. Belkin and P. Niyogi. Laplacian eigenmaps for dimensionality reduction and data representation. *Neural Comp.*, 15(6):1373–1396, 2003.

[2] C. M. Bishop, M. Svensen, and C. K. I. Williams. GTM: The generative topographic mapping. *Neural Computation*, 10:215–234, 1998.

[3] O. Chapelle, B. Schölkopf, and A. Zien, editors. *Semi-Supervised Learning*. MIT Press, Cambridge, 2006. in press, http://www.kyb.tuebingen.mpg.de/ssl-book.

[4] P. Grassberger and I. Procaccia. Measuring the strangeness of strange attractors. *Physica D*, 9:189–208, 1983.

[5] A. Grigoryan. Heat kernels on weighted manifolds and applications. *Cont. Math.*, 398:93–191, 2006.

[6] T. Hastie and W. Stuetzle. Principal curves. *J. Amer. Stat. Assoc.*, 84:502–516, 1989.

[7] M. Hein, J.-Y. Audibert, and U. von Luxburg. From graphs to manifolds - weak and strong pointwise consistency of graph Laplacians. In P. Auer and R. Meir, editors, *Proc. of the 18th Conf. on Learning Theory (COLT)*, pages 486–500, Berlin, 2005. Springer.

[8] M. Hein, J.-Y. Audibert, and U. von Luxburg. Graph Laplacians and their convergence on random neighborhood graphs, 2006. accepted at JMLR, available at arXiv:math.ST/0608522.

[9] M. Hein. *Geometrical aspects of statistical learning theory*. PhD thesis, MPI für biologische Kybernetik/Technische Universität Darmstadt, 2005.

[10] D. Keysers, W. Macherey, H. Ney, and J. Dahmen. Adaptation in statistical pattern recognition using tangent vectors. *IEEE Trans. on Pattern Anal. and Machine Intel.*, 26:269–274, 2004.

[11] C. Lange and K. Polthier. Anisotropic smoothing of point sets. *Computer Aided Geometric Design*, 22:680–692, 2005.

[12] O. Scherzer and J. Weickert. Relations between regularization and diffusion imaging. *J. of Mathematical Imaging and Vision*, 12:43–63, 2000.

[13] G. Taubin. A signal processing approach to fair surface design. In *Proc. of the 22nd annual conf. on Computer graphics and interactive techniques (Siggraph)*, pages 351–358, 1995.

[14] J. B. Tenenbaum, V. de Silva, and J. C. Langford. A global geometric framework for nonlinear dimensionality reduction. *Science*, 290(5500):2319–2323, 2000.

[15] D. Zhou, O. Bousquet, T. N. Lal, J. Weston, and B. Schölkopf. Learning with local and global consistency. In S. Thrun, L. Saul, and B. Schölkopf, editors, *Adv. in Neur. Inf. Proc. Syst. (NIPS)*, volume 16. MIT Press, 2004.
